# A Recurrent Neural Network Model of Velocity Storage in the Vestibulo-Ocular Reflex

**Thomas J. Anastasio**
Department of Otolaryngology
University of Southern California
School of Medicine
Los Angeles, CA 90033

## Abstract

A three-layered neural network model was used to explore the organization of the vestibulo-ocular reflex (VOR). The dynamic model was trained using recurrent back-propagation to produce compensatory, long duration eye muscle motoneuron outputs in response to short duration vestibular afferent head velocity inputs. The network learned to produce this response prolongation, known as velocity storage, by developing complex, lateral inhibitory interactions among the interneurons. These had the low baseline, long time constant, rectified and skewed responses that are characteristic of real VOR interneurons. The model suggests that all of these features are interrelated and result from lateral inhibition.

## 1 SIGNAL PROCESSING IN THE VOR

The VOR stabilizes the visual image by producing eye rotations that are nearly equal and opposite to head rotations (Wilson and Melvill Jones 1979). The VOR utilizes head rotational velocity signals, which originate in the semicircular canal receptors of the inner ear, to control contractions of the extraocular muscles. The reflex is coordinated by brainstem interneurons in the vestibular nuclei (VN), that relay signals from canal afferent sensory neurons to eye muscle motoneurons.

The VN interneurons, however, do more than just relay signals. Among other functions, the VN neurons process the canal afferent signals, stretching out their time constants by about four times before transmitting this signal to the motoneurons. This time constant prolongation, which is one of the clearest examples of signal processing in motor neurophysiology, has been termed velocity storage (Raphan et al. 1979). The neural mechanisms underlying velocity storage, however, remain unidentified.

The VOR is bilaterally symmetric (Wilson and Melvill Jones 1979). The semicircular canals operate in push-pull pairs, and the extraocular muscles are arranged in agonist/antagonist pairs. The VN are also arranged bilaterally and interact via inhibitory commissural connections. The commissures are necessary for velocity storage, which is eliminated by cutting the commissures in monkeys (Blair and Gavin 1981).

When the overall VOR fails to compensate for head rotations, the visual image is not stabilized but moves across the retina at a velocity that is equal to the amount of VOR error. This 'retinal slip' signal is transmitted back to the VN, and is known to modify VOR operation (Wilson and Melvill Jones 1979). Thus the VOR can be modeled beautifully as a three-layered neural network, complete with recurrent connections and error signal back-propagation at the VN level. By modeling the VOR as a neural network, insight can be gained into the global organization of this reflex.

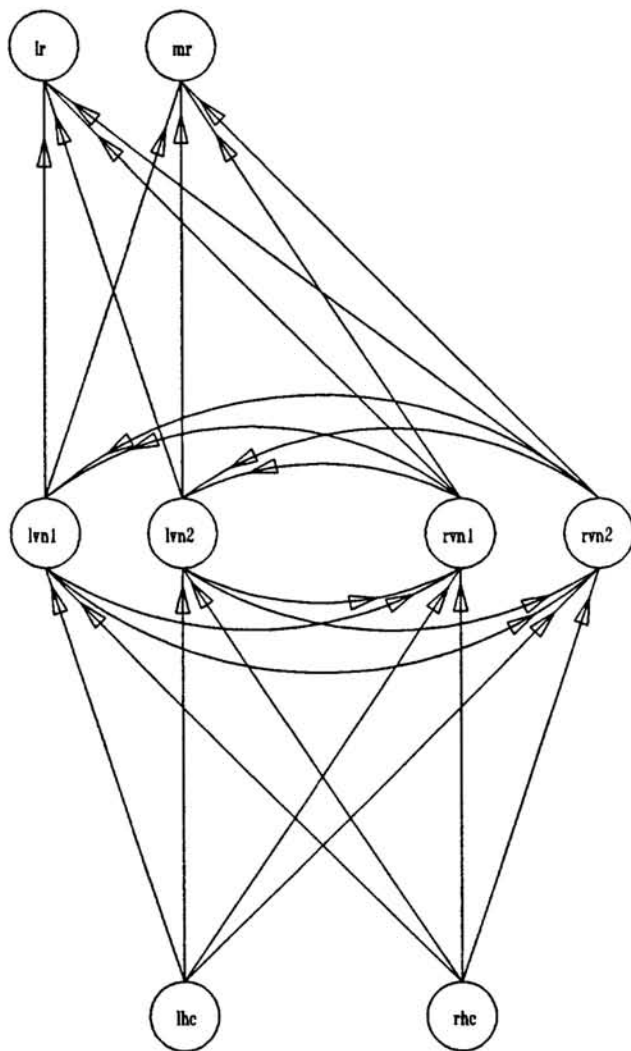

Figure 1: Architecture of the Horizontal VOR Neural Network Model. lhc and rhc, left and right horizontal canal afferents; lvn and rvn, left and right VN neurons; lr and mr, lateral and medial rectus motoneurons of the left eye. This and all subsequent figures are redrawn from Anastasio (1991), with permission.

## 2 ARCHITECTURE OF THE VOR NEURAL NETWORK MODEL

The recurrent neural network model of the horizontal VOR is diagrammed in Fig. 1. The input units represent afferents from the left and right horizontal semicircular canals (lhc and rhc). These are the canals and afferents that respond to yaw head rotations (as in shaking the head 'no'). The output units represent motoneurons of the lateral and medial rectus muscles of the left eye (lr and mr). These are the motoneurons and muscles that move the eye in the yaw plane. The units in the hidden layer correspond to interneurons in the VN, on both the left and right sides of the brainstem (lvn1, lvn2, rvn1 and rvn2). All units compute the weighted sum of their inputs and then pass this sum through the sigmoidal squashing function.

To represent the VOR relay, input project to hidden units and hidden project to output units. Commissural connections are modeled as lateral interconnections between hidden units on opposite sides of the brainstem. The model is constrained to allow only those connections that have been experimentally well described in mammals. For example, canal afferents do not project directly to motoneurons in mammals, and so direct connections from input to output units are not included in the model.

Evidence to date suggests that plastic modification of synapses may occur at the VN level but not at the motoneurons. The weights of synapses from hidden to output units are therefore fixed. All fixed hidden-to-output weights have the same absolute value, and are arranged in a reciprocal pattern. Hidden units lvn1 and lvn2 inhibit lr and excite mr; the opposite pattern obtains for rvn1 and rvn2. The connections to the hidden units, from input or contralateral hidden units, were initially randomized and then modified by the continually running, recurrent back-propagation algorithm of Williams and Zipser (1989).

## 3 TRAINING AND ANALYZING THE VOR NETWORK MODEL

The VOR neural network model was trained to produce compensatory motoneuron responses to two impulse head accelerations, one to the left and the other to the right, presented repeatedly in random order. The preset impulse responses of the canal afferents (input units) decay with a time constant of one network cycle or tick (Fig. 2, A and B, solid). The desired motoneuron (output unit) responses are equal and opposite in amplitude to the afferent responses, producing compensatory eye movements, but decay with a time constant four times longer, reflecting velocity storage (Fig. 2, A and B, dashed). Because of the three-layered architecture of the VOR, a delay of one network cycle is introduced between the input and output responses.

After about 5000 training set presentations, the network learned to match actual and desired output responses quite closely (Fig. 2, C and D, solid and dashed, respectively). The input-to-hidden connections arranged themselves in a reciprocal pattern, each input unit exciting the ipsilateral hidden units and inhibiting the contralateral ones. This arrangement is also observed for the actual VOR (Wilson and Melvill Jones 1979). The hidden-to-hidden (commissural) connections formed overlapping, lateral inhibitory feedback loops. These loops mediate velocity storage in the network. Their removal results in a loss of velocity storage (a decrease in output time constants from four to one tick), and also slightly increases output unit sensitivity (Fig. 2, C and D, dotted). These effects on VOR are also observed following commissurotomy in monkeys (Blair and Gavin 1981).

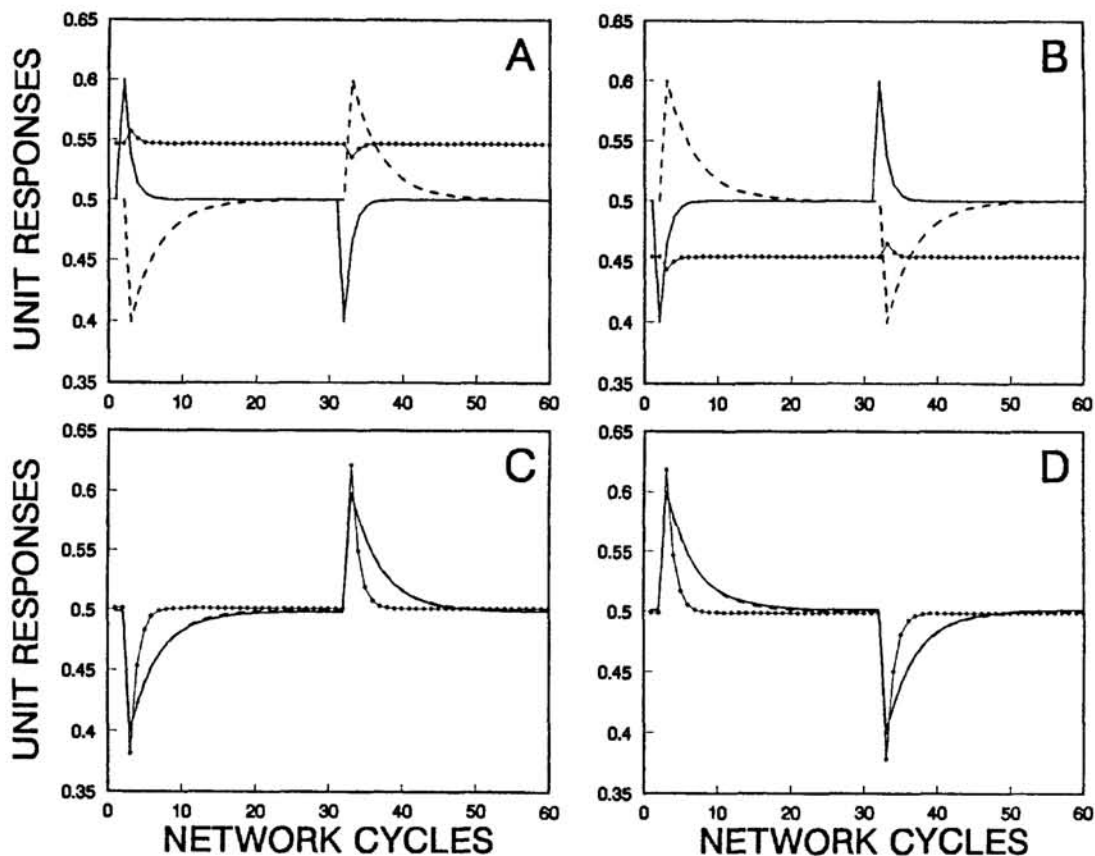

Figure 2: Training the VOR Network Model.  A and B, input unit responses (solid), desired output unit responses (dashed), and incorrect output responses of initially randomized network (dotted); lhc and lr in A, rhc and mr in B.  C and D, desired output responses (dashed), actual output responses of trained network (solid), and output responses following removal of commissural connections (dotted); lr in C, mr in D.

Although all the hidden units project equally strongly to the output units, the inhibitory connections between them, and their response patterns, are different.  Hidden units lvn1 and rvn1 have developed strong mutual inhibition.  Thus units lvn1 and rvn1 exert net positive feedback on themselves.  Their responses appear as low-pass filtered versions of the input unit responses (Fig. 3, A, solid and dashed).  In contrast, hidden units lvn2 and rvn2 have almost zero mutual inhibition, and tend to pass the sharply peaked input responses unaltered (Fig. 3, B, solid and dashed).  Thus the hidden units appear to form parallel integrated (lvn1 and rvn1) and direct (lvn2 and rvn2) pathways to the outputs.   This parallel arrangement for velocity storage was originally suggested by Raphan and coworkers (1979).  However, units lvn2 and rvn2 are coupled to units rvn1 and lvn1, respectively, with moderately strong mutual inhibition.  This coupling endows units lvn2 and rvn2 with longer overall decay times than they would have by themselves.  This arrangement resembles the mechanism of feedback through a neural low-pass filter, suggested by Robinson (1981) to account for velocity storage.  Thus, the network model gracefully combines the two mechanisms that have been identified for velocity storage, in what may be a more optimal configuration than either one alone.

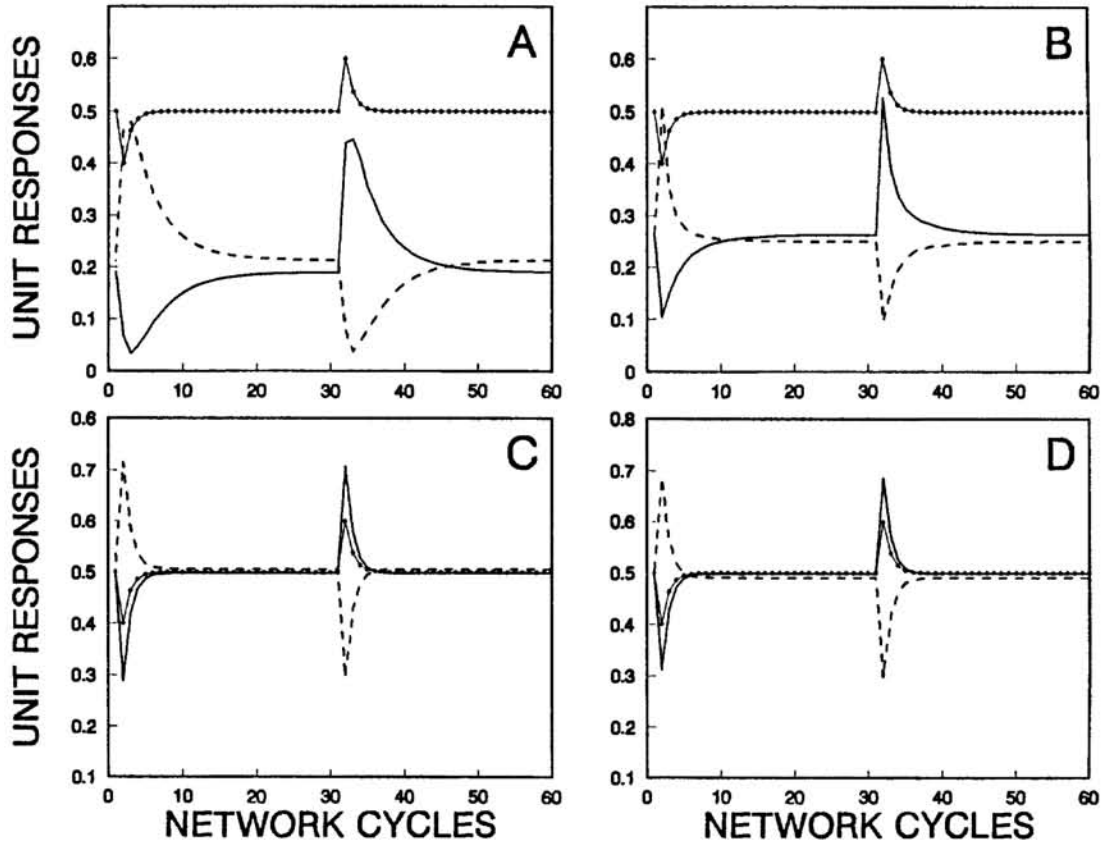

Figure 3: Responses of Model VN Interneurons. Networks trained with (A and B) and without (C and D) velocity storage. A and C, rvn1, solid; lvn1, dashed. B and D, rvn2, solid; lvn2, dashed. rhc, dotted, all plots.

Besides having longer time constants, the hidden units also have lower baseline firing rates and higher sensitivities than the input units (Fig. 3, A and B). The lower baseline forces the hidden units to operate closer to the bottom of the squashing function. This in turn causes the hidden units to have asymmetric responses, larger in the excitatory than in the inhibitory directions. Actual VN interneurons also have higher sensitivities, longer time constants, lower baseline firing rates and asymmetric responses as compared to canal afferents (Fuchs and Kimm 1975; Buettner et al. 1978).

For purposes of comparison, the network was retrained to produce a VOR without velocity storage (inputs and desired outputs had the same time constant of one tick). All of the hidden units in this network developed almost zero lateral inhibition. Although they also had higher sensitivities than the input units, their responses otherwise resembled input responses (Fig. 3, C and D). This demonstrates that the long time constant, low baseline and asymmetric responses of the hidden units are all interrelated by commissural inhibition in the network, which may be the case for actual VN interneurons as well.

# 4 NONLINEAR BEHAVIOR OF THE VOR NETWORK MODEL

Because hidden units have low baseline firing rates, larger inputs can produce inhibitory hidden unit responses that are forced into the low-sensitivity region of squashing function or even into cut-off. Hidden unit cut-off breaks the feedback loops that subserve velocity storage. This produces nonlinearities in the responses of the hidden and output units.

For example, an impulse input at twice the amplitude of the training input produces larger output unit responses (Fig. 4, A, solid), but these decay at a faster rate than expected (Fig. 4, A, dot-dash). Faster decay results because inhibitory hidden unit responses are cutting-off at the higher amplitude level (Fig. 4, C, solid). This cut-off disrupts velocity storage, decreasing the integrative properties of the hidden units (Fig. 4, C, solid) and increasing output unit decay rate.

Nonlinear responses are even more apparent with sinusoidal input. At low input levels, the output responses are also sinusoidal and their phase lag relative to the input is commensurate with their time constant of four ticks (Fig. 4, B, dashed). As sinusoidal in-

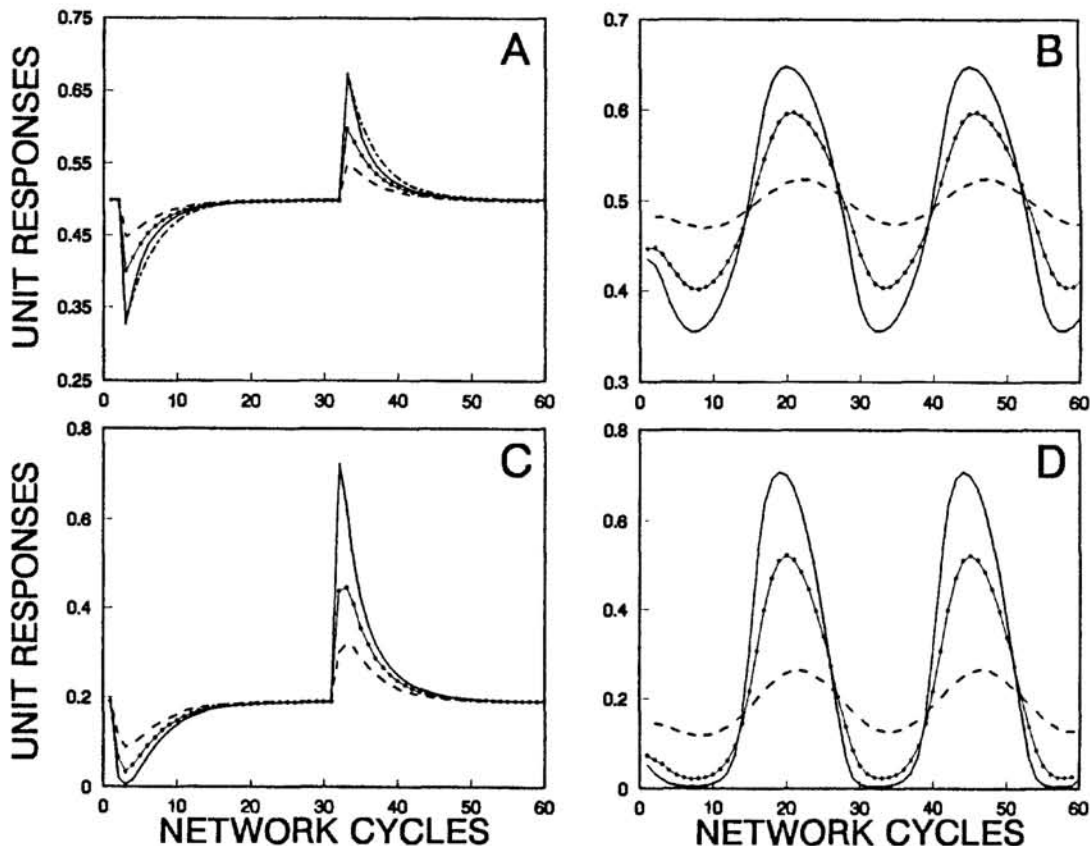

Figure 4: Nonlinear Responses of Model VOR Neurons. A and C, responses of lr (A) and rvn1 (C) to impulse inputs at low (dashed), medium (training, dotted) and high (solid) amplitudes. A, expected lr response at high input amplitude with time constant of four ticks (dot-dash). B and D, response of lr (B) and rvn1 (D) to sinusoidal inputs at low (dashed), medium (dotted) and high (solid) amplitudes.

put amplitude increases, however, output response phase lag decreases, signifying a decrease in time constant (Fig. 4, B, dotted and solid). Also, the output responses skew, such that the excursions from baseline are steeper than the returns. Time constant decrease and skewing with increase in head rotation amplitude are also characteristic of the VOR in monkeys (Paige 1983). Again, these nonlinearities are associated with hidden unit cut-off (Fig. 4, D, dotted and solid), which disrupts velocity storage, decreasing time constant and phase lag. Skewing results as the system time constant is lowered at peak and raised again midrange throughout each cycle of the responses. Actual VN neurons in monkeys exhibit similar cut-off (rectification) and skew (Fuchs and Kimm 1975; Buettner et al. 1978).

# 5 CONCLUSIONS

The VOR lends itself well to neural network modeling. The results summarized here, presented in detail elsewhere (Anastasio 1991), illustrate how neural network analysis can be used to study the organization of the VOR, and how its organization determines the response properties of the neurons that subserve this reflex.

**Acknowledgments**

This work was supported by the Faculty Research and Innovation Fund of the University of Southern California.

**References**

Anastasio, TJ (1991) Neural network models of velocity storage in the horizontal vestibulo-ocular reflex. Biol Cybern 64: 187-196

Blair SM, Gavin M (1981) Brainstem commissures and control of time constant of vestibular nystagmus. Acta Otolaryngol 91: 1-8

Buettner UW, Buttner U, Henn V (1978) Transfer characteristics of neurons in vestibular nuclei of the alert monkey. J Neurophysiol 41: 1614-1628

Fuchs AF, Kimm J (1975) Unit activity in vestibular nucleus of the alert monkey during horizontal angular acceleration and eye movement. J Neurophysiol 38: 1140-1161

Paige GC (1983) Vestibuloocular reflex and its interaction with visual following mechanisms in the squirrel monkey. I. Response characteristics in normal animals. J Neurophysiol 49: 134-151

Raphan Th, Matsuo V, Cohen B (1979) Velocity Storage in the vestibulo-ocular reflex arc (VOR). Exp Brain Res 35: 229-248

Robinson DA (1981) The use of control systems analysis in the neurophysiology of eye movements. Ann Rev Neurosci 4: 463-503

Williams RJ, Zipser D (1989) A learning algorithm for continually running fully recurrent neural networks. Neural Comp 1: 270-280

Wilson VJ, Melvill Jones G (1979) Mammalian Vestibular Physiology. Plenum Press, New York